# Dynamical Causal Learning

**David Danks**
Institute for Human & Machine Cognition
University of West Florida
Pensacola, FL 32501
ddanks@ai.uwf.edu

**Thomas L. Griffiths**
Department of Psychology
Stanford University
Stanford, CA 94305-2130
gruffydd@psych.stanford.edu

**Joshua B. Tenenbaum**
Department of Brain & Cognitive Sciences
MIT
Cambridge, MA 02139
jbt@mit.edu

## Abstract

Current psychological theories of human causal learning and judgment focus primarily on long-run predictions: two by estimating parameters of a causal Bayes nets (though for different parameterizations), and a third through structural learning. This paper focuses on people's short-run behavior by examining dynamical versions of these three theories, and comparing their predictions to a real-world dataset.

## 1 Introduction

Currently active quantitative models of human causal judgment for single (and sometimes multiple) causes include conditional $\Delta P$ [8], power PC [1], and Bayesian network structure learning [4], [9]. All of these theories have some normative justification, and all can be understood rationally in terms of learning causal Bayes nets. The first two theories assume a parameterization for a Bayes net, and then perform maximum likelihood parameter estimation. Each has been the target of numerous psychological studies (both confirming and disconfirming) over the past ten years. The third theory uses a Bayesian structural score, representing the log likelihood ratio in favor of the existence of a connection between the potential cause and effect pair. Recent work found that this structural score gave a generally good account, and fit data that could be fit by neither of the other two models [9].

To date, all of these models have addressed only the static case, in which judgments are made after observing all of the data (either sequentially or in summary format). Learning in the real world, however, also involves dynamic tasks, in which judgments are made after each trial (or small number). Experiments on dynamic tasks, and theories that model human behavior in them, have received surprisingly little attention in the psychological community. In this paper, we explore dynamical variants of each of the above learning models, and compare their results to a real data set (from [7]). We focus only on the case of one potential cause, due to space and theoretical constraints, and a lack of experimental data for the multivariate case.

## 2 Real-World Data

In the experiment on which we focus in this paper [7], people's stepwise acquisition curves were measured by asking people to determine whether camouflage makes a tank more or less likely to be destroyed. Subjects observed a sequence of cases in which the tank was either camouflaged or not, and destroyed or not. They were asked after every five cases to judge the causal strength of the camouflage on a $[-100, +100]$ scale, where $-100$ and $+100$ respectively correspond to the potential cause always preventing or producing the effect. The learning curves, constructed from average strength ratings, were:

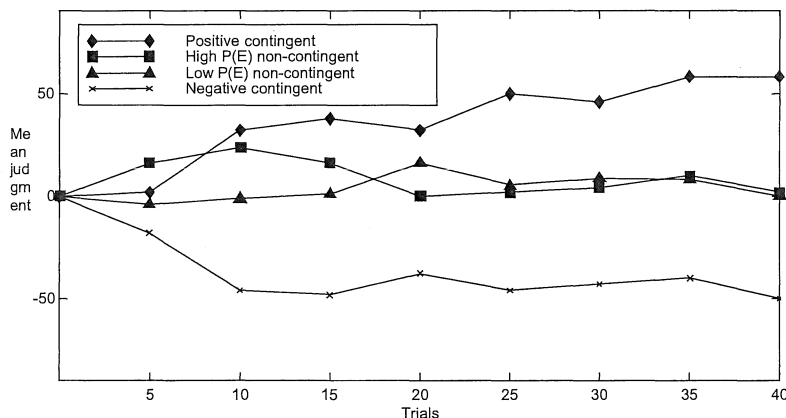

Figure 1: Example of learning curves

In this paper, we focus on qualitative features of the learning curves. These learning curves can be divided on the basis of the actual contingencies in the experimental condition. There were two *contingent* conditions: a positive condition in which $P(E \mid C) = .75$ (the probability of the effect given the cause) and $P(E \mid \neg C) = .25$, and a negative condition where the opposite was true. There were also two *non-contingent* conditions, one in which $P(E) = .75$ and one in which $P(E) = .25$, irrespective of the presence or absence of the causal variable. We refer to the former non-contingent condition as having a high $P(E)$, and the latter as having a low $P(E)$. There are two salient, qualitative features of the acquisition curves:

1. For contingent cases, the strength rating does not immediately reach the final judgment, but rather converges to it slowly; and

2. For non-contingent cases, there is an initial non-zero strength rating when the probability of the effect, $P(E)$, is high, followed by convergence to zero.

## 3 Parameter Estimation Theories

### 3.1 Conditional $\Delta P$

The conditional $\Delta P$ theory predicts that the causal strength rating for a particular factor will be (proportional to) the conditional contrast for that factor [5], [8]. The general form of the conditional contrast for a particular potential cause is given by: $\Delta P_{C.\{X\}} = P(E \mid C \ \& \ X) - P(E \mid \neg C \ \& \ X)$, where $X$ ranges over the possible states of the other potential causes. So, for example, if we have two potential causes, $C_1$ and $C_2$, then there are two conditional contrasts for $C_1$: $\Delta P_{C1.\{C2\}} = P(E \mid C_1 \ \& \ C_2) - P(E \mid \neg C_1 \ \& \ C_2)$ and $\Delta P_{C1.\{\neg C2\}} = P(E \mid C_1 \ \& \ \neg C_2) - P(E \mid \neg C_1 \ \& \ \neg C_2)$. Depending

on the probability distribution, some conditional contrasts for a potential cause may be undefined, and the defined contrasts for a particular variable may not agree. The conditional $\Delta P$ theory only makes predictions about a potential cause when the underlying probability distribution is "well-behaved": at least one of the conditional contrasts for the factor is defined, and all of the defined conditional contrasts for the factor are equal. For a single cause-effect relationship, calculation of the $\Delta P$ value is a maximum likelihood parameter estimator assuming that the cause and the background combine linearly to predict the effect [9].

Any long-run learning model can model sequential data by being applied to all of the data observed up to a particular point. That is, after observing $n$ datapoints, one simply applies the model, regardless of whether $n$ is "the long-run." The behavior of such a strategy for the conditional $\Delta P$ theory is shown in Figure 2 (a), and clearly fails to model accurately the above on-line learning curves. There is no gradual convergence to asymptote in the contingent cases, nor is there differential behavior in the non-contingent cases.

An alternative dynamical model is the Rescorla-Wagner model [6], which has essentially the same form as the well-known delta rule used for training simple neural networks. The R-W model has been shown to converge to the conditional $\Delta P$ value in exactly the situations in which the $\Delta P$ theory makes a prediction [2]. The R-W model follows a similar statistical logic as the $\Delta P$ theory: $\Delta P$ gives the maximum likelihood estimates in closed-form, and the R-W model essentially implements gradient ascent on the log-likelihood surface, as the delta rule has been shown to do. The R-W model produces learning curves that qualitatively fit the learning curves in Figure 1, but suffers from other serious flaws. For example, suppose a subject is presented with trials of $A$, $C$, and $E$, followed by trials with only $A$ and $E$. In such a task, called backwards blocking, the R-W model predicts that $C$ should be viewed as moderately causal, but human subjects rate $C$ as non-causal.

In the augmented R-W model [10] causal strength estimates (denoted by $V_i$, and assumed to start at zero) change after each observed case. Assuming that $\delta(X) = 1$ if $X$ occurs on a particular trial, and 0 otherwise, then strength estimates change by the following equation:

$$\Delta V_i = \alpha_{i\delta(C_i)}\beta_{\delta(E)}\left(\lambda\delta(E) - \sum_{\delta(C_j)=1} V_j\right).$$

$\alpha_{i0}$ and $\alpha_{i1}$ are rate parameters (saliences) applied when $C_i$ is present and absent, respectively, and $\beta_0$ and $\beta_1$ are the rate parameters when $E$ is present and absent, respectively. By updating the causal strengths of absent potential causes, this model is able to explain many of the phenomena that escape the normal R-W model, such as backwards blocking.

Although the augmented R-W model does not always have the same asymptotic behavior as the regular R-W model, it *does* have the same asymptotic behavior in exactly those situations in which the conditional $\Delta P$ theory makes a prediction (under typical assumptions: $\alpha_{i0} = -\alpha_{i1}$, $\beta_0 = \beta_1$, and $\lambda = 1$) [2]. To determine whether the augmented R-W model also captures the qualitative features of people's dynamic learning, we performed a simulation in which 1000 simulated individuals were shown randomly ordered cases that matched the probability distributions used in [7]. The model parameter values were $\lambda = 1.0$, $\alpha_{00} = 0.4$, $\alpha_{10} = 0.7$, $\alpha_{11} = -0.2$, $\beta_0 = \beta_1 = 0.5$, with two learned parameters: $V_0$ for the always present background cause $C_0$, and $V_1$ for the potential cause $C_1$. The mean values of $V_1$, multiplied by 100 to match scale with Figure 1, are shown in Figure 2 (b).

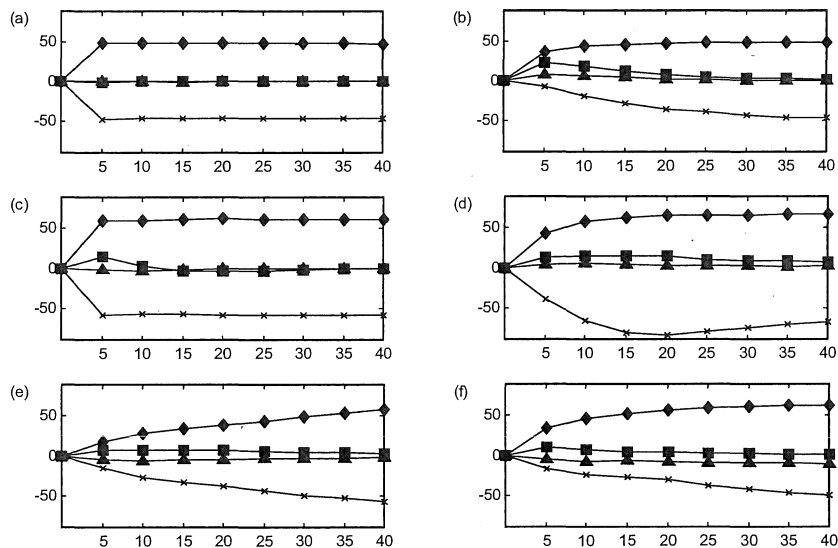

Figure 2: Modeling results. (a) is the maximum-likelihood estimate of $\Delta P$, (b) is the augmented R-W model, (c) is the maximum-likelihood estimate of causal power, (d) is the analogue of augmented R-W model for causal power, (e) shows the Bayesian strength estimate with a uniform prior on all parameters, and (f) does likewise with a beta(1,5) prior on $V_0$. The line-markers follow the conventions of Figure 1.

Variations in $\lambda$ only change the response scale. Higher values of $\alpha_{00}$ (the salience of the background) shift downward all early values of the learning curves, but do not affect the asymptotic values. The initial non-zero values for the non-contingent cases is proportional in size to $(\alpha_{10} + \alpha_{11})$, and so if the absence of the cause is more salient than the presence, the initial non-zero value will actually be negative. Raising the $\beta$ values increases the speed of convergence to asymptote, and the absolute values of the contingent asymptotes decrease in proportion to $(\beta_0 - \beta_1)$.

For the chosen parameter values, the learning curves for the contingent cases both gradually curve towards an asymptote, and in the non-contingent, high $P(E)$ case, there is an initial non-zero rating. Despite this qualitative fit and its computational simplicity, the augmented R-W model does not have a strong rational motivation. Its only rational justification is that it is a consistent estimator of $\Delta P$: in the limit of infinite data, it converges to $\Delta P$ under the same circumstances that the regular (and well-motivated) R-W model does. But it does not seem to have any of the other properties of a good statistical estimator: it is not unbiased, nor does it seem to be a maximum likelihood or gradient-ascent-on-log-likelihood algorithm (indeed, sometimes it appears to descend in likelihood). This raises the question of whether there might be an alternative dynamical model of causal learning that produces the appropriate learning curves but is also a principled, rational statistical estimator.

## 3.2 Power PC

In Cheng's power PC theory [1], causal strength estimates are predicted to be (proportional to) perceived causal power: the (unobserved) probability that the

potential cause, in the absence of all other causes, will produce the effect. Although causal power cannot be directly observed, it can be estimated from observed statistics given some assumptions. The power PC theory predicts that, when the assumptions are believed to be satisfied, causal power for (potentially) generative or preventive causes will be estimated by the following equations:

$$\text{Generative: } p_C = \frac{\Delta P_C}{1 - P(E \mid \neg C)} \qquad\qquad \text{Preventive: } p_C = \frac{-\Delta P_C}{P(E \mid \neg C)}$$

Because the power PC theory focuses on the long-run, one can easily determine which equation to use: simply wait until asymptote, determine $\Delta P_C$, and then divide by the appropriate factor. Similar equations can also be given for interactive causes. Note that although the preventive causal power equation yields a positive number, we should expect people to report a negative rating for preventive causes.

As with the $\Delta P$ theory, the power PC theory can, in the case of a single cause-effect pair, also be seen as a maximum likelihood estimator for the strength parameter of a causal Bayes net, though one with a different parameterization than for conditional $\Delta P$. Generative causes and the background interact to produce the effect as though they were a noisy-OR gate. Preventive causes combine with them as a noisy-AND-NOT gate. Therefore, if the $G_i$'s are generative causes and $I_j$'s are preventive causes, the theory predicts: $P(E) = \prod_j (1 - I_j) \left[ 1 - \prod_i (1 - G_i) \right].$

As for conditional $\Delta P$, simply applying the power PC equations to the sufficient statistics for observed sequential data does not produce appropriate learning curves. There is no gradual convergence in the contingent cases, and there is no initial difference in the non-contingent cases. This behavior is shown in Figure 2 (c).

Instead, we suggest using an analogue of the augmented R-W model, which uses the above noisy-OR/AND-NOT prediction instead of the linear prediction implicit in the augmented R-W model. Specifically, we define the following algorithm (with all parameters as defined before), using the notational device that the $C_k$'s are preventive and the $C_j$'s are generative:

$$\Delta V_i = \alpha_{i\delta(C_i)} \beta_{\delta(E)} \left( \lambda \delta(E) - \prod_{\delta(V_k)=1} (1 - V_k) \left[ 1 - \prod_{\delta(V_j)=1} (1 - V_j) \right] \right)$$

Unlike the R-W and augmented R-W models, there is no known characterization of the long-run behavior of this iterative algorithm. However, we can readily determine (using the equilibrium technique of [2]) the asymptotic $V_i$ values for one potential cause (and a single, always present, generative background cause). If we make the same simplifying assumptions as in Section 3.1, then this algorithm asymptotically computes the causal power for $C$, regardless of whether $C$ is generative or preventive. We conjecture that this algorithm also computes the causal power for multiple potential causes.

This iterative algorithm can only be applied if one knows whether each potential cause is potentially generative or preventive. Furthermore, we cannot determine directionality by the strategy of the power PC theory, as we do not necessarily have the correct $\Delta P$ sign during the short run. However, changing the classification of $C_i$ from generative to preventive (or *vice versa*) requires only removing from (adding to) the estimate (i) the $V_i$ term; and (ii) all terms in which $V_i$ was the only generative factor. Hence, we conjecture that this algorithm can be augmented to account for reclassification of potential causes after learning has begun.

To simulate this dynamical version of the power PC theory, we used the same setup as in Section 3.1 (and multiplied preventive causal power ratings by −1 to properly scale them). The parameters for this run were: $\lambda = 1.0$, $\alpha_{00} = 0.1$, $\alpha_{10} = 0.5$, $\alpha_{11} = -0.4$, $\beta_0 = \beta_1 = 0.9$, and the results are shown in Figure 2 (d). Parameter variations have the same effects as for the augmented R-W model, except that increasing $\alpha_{00}$ reduces the size of the initial non-zero values in the non-contingent conditions (instead of all conditions), and absolute values of the asymptotes in *all* conditions are shifted by an amount proportional to $(\beta_0 - \beta_1)$.

This dynamical theory produces the right sort of learning curves for these parameter values, and is also a consistent estimator (converging to the power PC estimate in the limit of infinite data). But as with the augmented R-W model, there is no rational motivation for choosing this dynamic estimator: it is not unbiased, nor maximum likelihood, nor an implementation of gradient ascent in log-likelihood. The theory's main (and arguably only) advantage over the augmented R-W model is that it converges to a quantity that is more typically what subjects estimate in long-run experiments. But it is still not what we desire from a principled dynamic model.

## 4   Bayesian structure learning

The learning algorithms considered thus far are based upon the idea that human causal judgments reflect the estimated value of a strength parameter in a particular (assumed) causal structure. Simple maximum likelihood estimation of these strength parameters does not capture the trends in the data, and so we have considered estimation algorithms that do not have a strong rational justification. We are thus led to the question of whether human learning curves *can* be accounted for by a rational process. In this section, we argue that the key to forming a rational, statistical explanation of people's dynamical behavior is to take structural uncertainty into account when forming parameter estimates.

Complete specification of the structure of a Bayesian network includes both the underlying graph and choice of parameterization. For example, in the present task there are three possible relationships between a potential cause $C_1$ and an effect $E$: generative ($h_+$), preventive ($h_-$), or non-existent ($h_0$). These three possibilities can respectively be represented by a graph with a noisy-OR parameterization, one with a noisy-AND-NOT parameterization, and one with no edge between the potential cause and the effect. Each possibility is illustrated schematically in Figure 3.

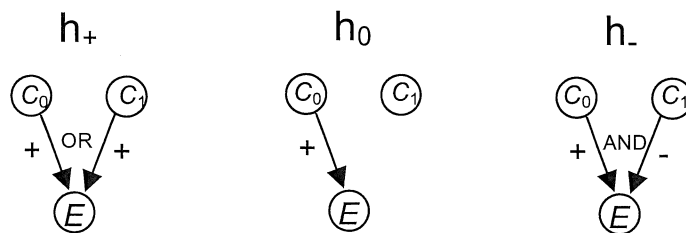

Figure 3: Structural hypotheses for the Bayesian model. $C_0$ is an always present background cause, $C_1$ is the potential cause, and $E$ the effect. The signs of arrows indicate positive and negative influences on the outcome.

Previous work applying Bayesian structure learning to human causal judgment focused on people making the decision as to which of these structures best accounts for the observed data [9]. That work showed that the likelihood of finding a causal relationship rose with the base rate $P(E)$ in non-contingent cases, suggesting that structural decisions are a relevant part of the present data. However, the rating scale

of the current task seems to encourage strength judgments rather than purely structural decisions, because it is anchored at the endpoints by two qualitatively different causal strengths (strong generative, strong preventive). As a result, subjects' causal judgments appear to converge to causal power.

Real causal learning tasks often involve uncertainty about both structure and parameters. Thus, even when a task demands ratings of causal strength, the structural uncertainty should still be taken into account; we do this by considering a hierarchy of causal models. The first level of this hierarchy involves structural uncertainty, giving equal probability to the relationship between the variables being generative, preventive, or non-existent. As mentioned in previous sections, the parameterizations associated with the first two models lead to a maximum likelihood estimate of causal power. The second level of the hierarchy addresses uncertainty over the parameters. With a constant background and a single cause, there are two parameters for the noisy-OR and the noisy-AND-NOT models, $V_0$ and $V_1$. If the cause and effect are unconnected, then only $V_0$ is required. Uncertainty in all parameters can be expressed with distributions on the unit interval.

Using this set of models, we can obtain a strength rating by taking the expectation of the strength parameter $V_i$ associated with a causal variable over the posterior distribution on that parameter induced by the data. This expectation is taken over both structure and parameters, allowing both factors to influence the result. In the two-variable case, we can write this as

$$<V_1> = \sum_{h \in H} \int_0^1 V_1 \, p(V_1 | h, D) \, p(h | D) dV_1$$

where $H = \{h_+, h_0, h_-\}$. The effective value of the strength parameter is 0 in the model where there is no relationship between cause and effect, and should be negative for preventive causes. We thus have:

$$<V_1> = P(h_+)\mu_+ - P(h_-)\mu_-$$

where $\mu_+$, $\mu_-$ are the posterior means of $V_1$ under $h_+$ and $h_-$ respectively.

While this theory is appealing from a rational and statistical point of view, it has computational drawbacks. All four terms in the above expression are quite computationally intensive to compute, and require an amount of information that increases exponentially with the number of causes. Furthermore, the number of different hypotheses we must consider grows exponentially with the number of potential causes, limiting its applicability for multivariate cases.

We applied this model to the data of [7], using a uniform prior over models, and also over parameters. The results, averaged across 200 random orderings of trials, are shown in Figure 2 (e). The predictions are somewhat symmetric with respect to positive and negative contingencies and high and low $P(E)$. This symmetry is a consequence of choosing a uniform (i.e., strongly uninformative) prior for the parameters. If we instead take a uniform prior on $V_1$ and a beta(1,5) prior on $V_0$, consistent with a prior belief that effects occur only rarely without an observed cause and similar to starting with zero weights in the algorithms presented above, we obtain the results shown in Figure 2 (f). In both cases, the curvature of the learning curves is a consequence of structural uncertainty, and the asymptotic values reflect the strength of causal relationships. In the contingent cases, the probability distribution over structures rapidly transfers all of its mass to the correct hypothesis, and the result asymptotes at the posterior mean of $V_1$ in that model, which will be very close to causal power. The initial non-zero ratings in the non-contingent cases result from $h_+$ giving a slightly better account of the data than $h_-$, essentially due to the non-uniform prior on $V_0$.

This structural account is only one means of understanding the rational basis for these learning curves. Dayan and Kakade [3] provide a statistical theory of classical conditioning based on Bayesian estimation of the parameters in a linear model similar to that underlying $\Delta P$. Their theory accounts for phenomena that the classical R-W theory does not, such as backwards blocking. They also give a neural network learning model that approximates the Bayesian estimate, and that closely resembles the augmented R-W model considered here. Their network model can also produce the learning curves discussed in this paper. However, because it is based on a linear model of causal interaction, it is not a good candidate for modeling human causal judgments, which across various studies of asymptotic behavior seem to be more closely approximated by parameter estimates in noisy logic gates, as instantiated in the power PC model [1] and our Bayesian model.

## 5  Conclusion

In this paper, we have outlined a range of dynamical models, from computationally simple ones (such as simply applying conditional $\Delta P$ to the observed datapoints) to rationally grounded ones (such as Bayesian structure/parameter estimation). Moreover, there seems to be a tension in this domain in trying to develop a model that is easily implemented in an individual and scales well with additional variables, and one that has a rational statistical basis. Part of our effort here has been aimed at providing a set of models that seem to equally well explain human behavior, but that have different virtues besides their fit with the data. Human causal learning might not scale up well, or it might not be rational; further discrimination among these possible theories awaits additional data about causal learning curves.

## References

[1] Cheng, Patricia W. 1997. "From Covariation to Causation: A Causal Power Theory." *Psychological Review*, 104 (2): 367-405.

[2] Danks, David. Forthcoming. "Equilibria of the Rescorla-Wagner Model." *Journal of Mathematical Psychology*.

[3] Dayan, Peter, & Kakade, Sham. 2001. "Explaining Away in Weight Space." In *Advances in Neural Information Processing Systems 13*.

[4] Gopnik, Alison, Clark Glymour, David M. Sobel, Laura E. Schulz, Tamar Kushnir, & David Danks. 2002. "A Theory of Causal Learning in Children: Causal Maps and Bayes Nets." Submitted to *Psychological Review*.

[5] Lober, Klaus, & David R. Shanks. 2000. "Is Causal Induction Based on Causal Power? Critique of Cheng (1997)." *Psychological Review*, 107 (1): 195-212.

[6] Rescorla, Robert A., & Allan R. Wagner. 1972. "A Theory of Pavlovian Conditioning: Variations in the Effectiveness of Reinforcement and Nonreinforcement." In A. H. Black & W. F. Prokasy, eds. *Classical Conditioning II: Current Research and Theory*. New York: Appleton-Century-Crofts. pp. 64-99.

[7] Shanks, David R. 1995. "Is Human Learning Rational?" *The Quarterly Journal of Experimental Psychology*, 48A (2): 257-279.

[8] Spellman, Barbara A. 1996. "Conditionalizing Causality." In D. R. Shanks, K. J. Holyoak, & D. L. Medin, eds. 1996. *Causal Learning: The Psychology of Learning and Motivation, Vol. 34*. San Diego, Calif.: Academic Press. pp. 167-206.

[9] Tenenbaum, Joshua B., & Thomas L. Griffiths. 2000. "Structure Learning in Human Causal Induction." In *Advances in Neural Information Processing Systems 13*.

[10] Van Hamme, Linda J., & Edward A. Wasserman. 1994. "Cue Competition in Causality Judgments: The Role of Nonpresentation of Compound Stimulus Elements." *Learning and Motivation*, 25: 127-151.
